# Lower Bounds on Rate of Convergence of Cutting Plane Methods

**Xinhua Zhang**
Dept. of Computing Science
University of Alberta
xinhua2@ualberta.ca

**Ankan Saha**
Dept. of Computer Science
University of Chicago
ankans@cs.uchicago.edu

**S.V.N. Vishwanathan**
Dept. of Statistics and
Dept. of Computer Science
Purdue University
vishy@stat.purdue.edu

## Abstract

In a recent paper Joachims [1] presented SVM-Perf, a cutting plane method (CPM) for training linear Support Vector Machines (SVMs) which converges to an $\epsilon$ accurate solution in $O(1/\epsilon^2)$ iterations. By tightening the analysis, Teo et al. [2] showed that $O(1/\epsilon)$ iterations suffice. Given the impressive convergence speed of CPM on a number of practical problems, it was conjectured that these rates could be further improved. In this paper we disprove this conjecture. We present counter examples which are not only applicable for training linear SVMs with hinge loss, but also hold for support vector methods which optimize a *multivariate* performance score. However, surprisingly, these problems are not inherently hard. By exploiting the structure of the objective function we can devise an algorithm that converges in $O(1/\sqrt{\epsilon})$ iterations.

## 1 Introduction

There has been an explosion of interest in machine learning over the past decade, much of which has been fueled by the phenomenal success of binary Support Vector Machines (SVMs). Driven by numerous applications, recently, there has been increasing interest in support vector learning with linear models. At the heart of SVMs is the following regularized risk minimization problem:

$$\min_{\mathbf{w}} J(\mathbf{w}) := \underbrace{\frac{\lambda}{2} \|\mathbf{w}\|^2}_{\text{regularizer}} + \underbrace{R_{\text{emp}}(\mathbf{w})}_{\text{empirical risk}} \quad \text{with} \quad R_{\text{emp}}(\mathbf{w}) := \frac{1}{n} \sum_{i=1}^{n} \max(0, 1 - y_i \langle \mathbf{w}, \mathbf{x}_i \rangle). \quad (1)$$

Here we assume access to a training set of $n$ labeled examples $\{(\mathbf{x}_i, y_i)\}_{i=1}^{n}$ where $\mathbf{x}_i \in \mathbb{R}^d$ and $y_i \in \{-1, +1\}$, and use the square Euclidean norm $\|\mathbf{w}\|^2 = \sum_i w_i^2$ as the regularizer. The parameter $\lambda$ controls the trade-off between the empirical risk and the regularizer.

There has been significant research devoted to developing specialized optimizers which minimize $J(\mathbf{w})$ efficiently. In an award winning paper, Joachims [1] presented a cutting plane method (CPM)[1], SVM-Perf, which was shown to converge to an $\epsilon$ accurate solution of (1) in $O(1/\epsilon^2)$ iterations, with each iteration requiring $O(nd)$ effort. This was improved by Teo et al. [2] who showed that their Bundle Method for Regularized Risk Minimization (BMRM) (which encompasses SVM-Perf as a special case) converges to an $\epsilon$ accurate solution in $O(nd/\epsilon)$ time.

While online learning methods are becoming increasingly popular for solving (1), a key advantage of CPM such as SVM-Perf and BMRM is their ability to *directly* optimize nonlinear multivariate performance measures such as $F_1$-score, ordinal regression loss, and ROCArea which are widely used in some application areas. In this case $R_{\text{emp}}$ does not decompose into a sum of losses over individual data points like in (1), and hence one has to employ batch algorithms. Letting $\Delta(\mathbf{y}, \bar{\mathbf{y}})$ denote the multivariate discrepancy between the correct labels $\mathbf{y} := (y_1, \ldots, y_n)^\top$ and a candidate labeling $\bar{\mathbf{y}}$ (to be concretized later), the $R_{\text{emp}}$ for the multivariate measure is formulated by [3] as

$$R_{\text{emp}}(\mathbf{w}) = \max_{\bar{\mathbf{y}} \in \{-1,1\}^n} \left[ \Delta(\mathbf{y}, \bar{\mathbf{y}}) + \frac{1}{n} \sum_{i=1}^{n} \langle \mathbf{w}, \mathbf{x}_i \rangle \, (\bar{y}_i - y_i) \right]. \tag{2}$$

In another award winning paper by Joachims [3], the regularized risk minimization problems corresponding to these measures are optimized by using a CPM.

Given the widespread use of CPM in machine learning, it is important to understand their convergence guarantees in terms of the upper and lower bounds on the number of iterations needed to converge to an $\epsilon$ accurate solution. The tightest, $O(1/\epsilon)$, upper bounds on the convergence speed of CPM is due to Teo et al. [2], who analyzed a restricted version of BMRM which only optimizes over one dual variable per iteration. However, on practical problems the observed rate of convergence is significantly faster than predicted by theory. Therefore, it had been conjectured that the upper bounds might be further tightened via a more refined analysis. In this paper we construct counter examples for both decomposable $R_{\text{emp}}$ like in equation (1) and non-decomposable $R_{\text{emp}}$ like in equation (2), on which CPM requires $\Omega(1/\epsilon)$ iterations to converge, thus disproving this conjecture[2]. We will work with BMRM as our prototypical CPM. As Teo et al. [2] point out, BMRM includes many other CPM such as SVM-Perf as special cases.

Our results lead to the following natural question: Do the lower bounds hold because regularized risk minimization problems are fundamentally hard, or is it an inherent limitation of CPM? In other words, to solve problems such as (1), does there exist a solver which requires less than $O(nd/\epsilon)$ effort (better in $n, d$ *and* $\epsilon$)? We provide partial answers. To understand our contribution one needs to understand the two standard assumptions that are made when proving convergence rates:

- **A1:** The data points $\mathbf{x}_i$ lie inside a $L_2$ (Euclidean) ball of radius $R$, that is, $\|\mathbf{x}_i\| \le R$.
- **A2:** The subgradient of $R_{\text{emp}}$ is bounded, *i.e.*, at any point $\mathbf{w}$, there exists a subgradient $\mathbf{g}$ of $R_{\text{emp}}$ such that $\|\mathbf{g}\| \le G < \infty$.

Clearly assumption **A1** is more restrictive than **A2**. By adapting a result due to [6] we show that one can devise an $O(nd/\sqrt{\epsilon})$ algorithm for the case when assumption **A1** holds. Finding a fast optimizer under assumption **A2** remains an open problem.

**Notation:** Lower bold case letters (*e.g.*, $\mathbf{w}$, $\boldsymbol{\mu}$) denote vectors, $w_i$ denotes the $i$-th component of $\mathbf{w}$, $\mathbf{0}$ refers to the vector with all zero components, $\mathbf{e}_i$ is the $i$-th coordinate vector (all 0's except 1 at the $i$-th coordinate) and $\Delta_k$ refers to the $k$ dimensional simplex. Unless specified otherwise, $\langle \cdot, \cdot \rangle$ denotes the Euclidean dot product $\langle \mathbf{x}, \mathbf{w} \rangle = \sum_i x_i w_i$, and $\|\cdot\|$ refers to the Euclidean norm $\|\mathbf{w}\| := (\langle \mathbf{w}, \mathbf{w} \rangle)^{1/2}$. We denote $\overline{\mathbb{R}} := \mathbb{R} \cup \{\infty\}$, and $[t] := \{1, \dots, t\}$.

Our paper is structured as follows. We briefly review BMRM in Section 2. Two types of lower bounds are subsequently defined in Section 3, and Section 4 contains descriptions of various counter examples that we construct. In Section 5 we describe an algorithm which provably converges to an $\epsilon$ accurate solution of (1) in $O(1/\sqrt{\epsilon})$ iterations under assumption **A1**. The paper concludes with a discussion and outlook in Section 6. Technical proofs and a ready reckoner of the convex analysis concepts used in the paper can be found in [7, Appendix A].

## 2 BMRM

At every iteration, BMRM replaces $R_{\text{emp}}$ by a piecewise linear lower bound $R_k^{\text{cp}}$ and optimizes [2]

$$\min_{\mathbf{w}} J_k(\mathbf{w}) := \frac{\lambda}{2} \|\mathbf{w}\|^2 + R_k^{\text{cp}}(\mathbf{w}), \quad \text{where } R_k^{\text{cp}}(\mathbf{w}) := \max_{1 \le i \le k} \langle \mathbf{w}, \mathbf{a}_i \rangle + b_i, \tag{3}$$

to obtain the next iterate $\mathbf{w}_k$. Here $\mathbf{a}_i \in \partial R_{\text{emp}}(\mathbf{w}_{i-1})$ denotes an arbitrary subgradient of $R_{\text{emp}}$ at $\mathbf{w}_{i-1}$ and $b_i = R_{\text{emp}}(\mathbf{w}_{i-1}) - \langle \mathbf{w}_{i-1}, \mathbf{a}_i \rangle$. The piecewise linear lower bound is successively tightened until the gap

$$\epsilon_k := \min_{0 \le t \le k} J(\mathbf{w}_t) - J_k(\mathbf{w}_k) \tag{4}$$

falls below a predefined tolerance $\epsilon$.

Since $J_k$ in (3) is a convex objective function, one can compute its dual. Instead of minimizing $J_k$ with respect to $\mathbf{w}$ one can equivalently maximize the dual [2] over the $k$ dimensional simplex:

$$D_k(\boldsymbol{\alpha}) = -\frac{1}{2\lambda} \|A_k \boldsymbol{\alpha}\|^2 + \langle \mathbf{b}_k, \boldsymbol{\alpha} \rangle, \quad \text{where} \quad \boldsymbol{\alpha} \in \Delta_k, \tag{5}$$

| **Algorithm 1:** qp-bmrm: solving the inner loop of BMRM exactly via full QP. | **Algorithm 2:** ls-bmrm: solving the inner loop of BMRM approximately via line search. |
|---|---|
| **Require:** Previous subgradients $\{\mathbf{a}_i\}_{i=1}^k$ and intercepts $\{b_i\}_{i=1}^k$. | **Require:** Previous subgradients $\{\mathbf{a}_i\}_{i=1}^k$ and intercepts $\{b_i\}_{i=1}^k$. |
| 1: Set $A_k := (\mathbf{a}_1, \ldots, \mathbf{a}_k)$, $\mathbf{b}_k := (b_1, \ldots b_k)^\top$. | 1: Set $A_k := (\mathbf{a}_1, \ldots, \mathbf{a}_k)$, $\mathbf{b}_k := (b_1, \ldots b_k)^\top$. |
| | 2: Set $\boldsymbol{\alpha}(\eta) := \left(\eta\boldsymbol{\alpha}_{k-1}^\top, 1-\eta\right)^\top$. |
| 2: $\boldsymbol{\alpha}_k \leftarrow \underset{\boldsymbol{\alpha}\in\Delta_k}{\operatorname{argmax}}\left\{-\frac{1}{2\lambda}\|A_k\boldsymbol{\alpha}\|^2 + \langle\boldsymbol{\alpha}, \mathbf{b}_k\rangle\right\}$. | 3: $\eta_k \leftarrow \underset{\eta\in[0,1]}{\operatorname{argmax}}\left\{\frac{-1}{2\lambda}\|A_k\boldsymbol{\alpha}(\eta)\|^2 + \langle\boldsymbol{\alpha}(\eta), \mathbf{b}_k\rangle\right\}$. |
| | 4: $\boldsymbol{\alpha}_k \leftarrow \left(\eta_k\boldsymbol{\alpha}_{k-1}^\top, 1-\eta_k\right)^\top$. |
| 3: **return** $\mathbf{w}_k = -\lambda^{-1}A_k\boldsymbol{\alpha}_k$. | 5: **return** $\mathbf{w}_k = -\lambda^{-1}A_k\boldsymbol{\alpha}_k$. |

and set $\boldsymbol{\alpha}_k = \operatorname{argmax}_{\boldsymbol{\alpha}\in\Delta_k} D_k(\boldsymbol{\alpha})$. Note that $A_k$ and $\mathbf{b}_k$ in (5) are defined in Algorithm 1. Since maximizing $D_k(\boldsymbol{\alpha})$ is a quadratic programming (QP) problem, we call this algorithm qp-bmrm. Pseudo-code can be found in Algorithm 1.

Note that at iteration $k$ the dual $D_k(\boldsymbol{\alpha})$ is a QP with $k$ variables. As the number of iterations increases the size of the QP also increases. In order to avoid the growing cost of the dual optimization at each iteration, [2] proposed using a one-dimensional line search to calculate an approximate maximizer $\boldsymbol{\alpha}_k$ on the line segment $\{(\eta\boldsymbol{\alpha}_{k-1}^\top, (1-\eta))^\top : \eta \in [0,1]\}$, and we call this variant ls-bmrm. Pseudo-code can be found in Algorithm 2. We refer the reader to [2] for details.

Even though qp-bmrm solves a more expensive optimization problem $D_k(\boldsymbol{\alpha})$ per iteration, Teo et al. [2] could only show that both variants of BMRM converge at $O(1/\epsilon)$ rates:

**Theorem 1 ([2])** *Suppose assumption* **A2** *holds. Then for any* $\epsilon < 4G^2/\lambda$, *both* ls-bmrm *and* qp-bmrm *converge to an* $\epsilon$ *accurate solution of* (1) *as measured by* (4) *after at most the following number of steps:*
$$\log_2 \frac{\lambda J(\mathbf{0})}{G^2} + \frac{8G^2}{\lambda\epsilon} - 1.$$

**Generality of BMRM** Thanks to the formulation in (3) which only uses $R_{\text{emp}}$, BMRM is applicable to a wide variety of $R_{\text{emp}}$. For example, when used to train binary SVMs with $R_{\text{emp}}$ specified by (1), it yields exactly the SVM-Perf algorithm [1]. When applied to optimize the multivariate score, *e.g.* $F_1$-score with $R_{\text{emp}}$ specified by (2), it immediately leads to the optimizer given by [3].

## 3  Upper and Lower Bounds

Since most rates of convergence discussed in the machine learning community are upper bounds, it is important to rigorously define the meaning of a lower bound with respect to $\epsilon$, and to study its relationship with the upper bounds. At this juncture it is also important to clarify an important technical point. Instead of minimizing the objective function $J(\mathbf{w})$ defined in (1), if we minimize a scaled version $cJ(\mathbf{w})$ this scales the approximation gap (4) by $c$. Assumptions such as **A1** and **A2** fix this degree of freedom by bounding the scale of the objective function.

Given a function $f \in \mathcal{F}$ and an optimization algorithm $A$, suppose $\{\mathbf{w}_k\}$ are the iterates produced by the algorithm $A$ when minimizing $f$. Define $T(\epsilon; f, A)$ as the first step index $k$ when $\mathbf{w}_k$ becomes an $\epsilon$ accurate solution[3]:
$$T(\epsilon; f, A) = \min\left\{k : f(\mathbf{w}_k) - \min_{\mathbf{w}} f(\mathbf{w}) \le \epsilon\right\}. \tag{6}$$

Upper and lower bounds are both properties for a pair of $\mathcal{F}$ and $A$. A function $g(\epsilon)$ is called *an upper bound of* $(\mathcal{F}, A)$ if for all functions $f \in \mathcal{F}$ and all $\epsilon > 0$, it takes at most order $g(\epsilon)$ steps for $A$ to reduce the gap to less than $\epsilon$, *i.e.*,
$$\textbf{(UB)} \qquad \forall \epsilon > 0, \forall f \in \mathcal{F}, \ T(\epsilon; f, A) \le g(\epsilon). \tag{7}$$

On the other hand, lower bounds can be defined in two different ways depending on how the above two universal qualifiers are flipped to existential qualifiers.

| Algorithms | Assuming **A1** | | | Assuming **A2** | | |
|---|---|---|---|---|---|---|
| | UB | SLB | WLB | UB | SLB | WLB |
| ls-bmrm | $O(1/\epsilon)$ | $\Omega(1/\epsilon)$ | $\Omega(1/\epsilon)$ | $O(1/\epsilon)$ | $\Omega(1/\epsilon)$ | $\Omega(1/\epsilon)$ |
| qp-bmrm | $O(1/\epsilon)$ | open | open | $O(1/\epsilon)$ | open | $\Omega(1/\epsilon)$ |
| Nesterov | $O(1/\sqrt{\epsilon})$ | $\Omega(1/\sqrt{\epsilon})$ | $\Omega(1/\sqrt{\epsilon})$ | $n/a$ | $n/a$ | $n/a$ |

Table 1: Summary of the known upper bounds and our lower bounds. Note: **A1** $\Rightarrow$ **A2**, but not vice versa. SLB $\Rightarrow$ WLB, but not vice versa. UB is tight, if it matches WLB.

- **Strong lower bounds (SLB)** $h(\epsilon)$ is called *a* SLB of $(\mathcal{F}, A)$ if there exists a function $\tilde{f} \in \mathcal{F}$, such that for all $\epsilon > 0$ it takes at least $h(\epsilon)$ steps for $A$ to find an $\epsilon$ accurate solution of $\tilde{f}$:

$$\textbf{(SLB)} \qquad \exists\, \tilde{f} \in \mathcal{F},\ s.t.\ \forall\, \epsilon > 0,\ \ T(\epsilon; \tilde{f}, A) \geq h(\epsilon). \qquad (8)$$

- **Weak lower bound (WLB)** $h(\epsilon)$ is called *a* WLB of $(\mathcal{F}, A)$ if for any $\epsilon > 0$, there exists a function $f_\epsilon \in \mathcal{F}$ depending on $\epsilon$, such that it takes at least $h(\epsilon)$ steps for $A$ to find an $\epsilon$ accurate solution of $f_\epsilon$:

$$\textbf{(WLB)} \qquad \forall\, \epsilon > 0, \exists\, f_\epsilon \in \mathcal{F},\ s.t.\ \ T(\epsilon; f_\epsilon, A) \geq h(\epsilon). \qquad (9)$$

Clearly, the existence of a SLB implies a WLB. However, it is usually much harder to establish SLB than WLB. Fortunately, WLBs are sufficient to refute upper bounds or to establish their tightness. The size of the function class $\mathcal{F}$ affects the upper and lower bounds in opposite ways. Suppose $\mathcal{F}' \subset \mathcal{F}$. Proving upper (resp. lower) bounds on $(\mathcal{F}', A)$ is usually easier (resp. harder) than proving upper (resp. lower) bounds for $(\mathcal{F}, A)$.

## 4 Constructing Lower Bounds

Letting the minimizer of $J(\mathbf{w})$ be $\mathbf{w}^*$, we are interested in bounding the *primal gap* of the iterates $\mathbf{w}_k : J(\mathbf{w}_k) - J(\mathbf{w}^*)$. Datasets will be constructed explicitly whose resulting objective $J(\mathbf{w})$ will be shown to attain the lower bounds of the algorithms. The $R_{\mathrm{emp}}$ for both the hinge loss in (1) and the $F_1$-score in (2) will be covered, and our results are summarized in Table 1. Note that as assumption **A1** implies **A2** and SLB implies WLB, some entries of the table imply others.

### 4.1 Strong Lower Bounds for Solving Linear SVMs using ls-bmrm

We first prove the $\Omega(1/\epsilon)$ lower bound for ls-bmrm on SVM problems under assumption **A1**. Consider a one dimensional training set with four examples: $(x_1, y_1) = (-1, -1)$, $(x_2, y_2) = (-\frac{1}{2}, -1)$, $(x_3, y_3) = (\frac{1}{2}, 1)$, $(x_4, y_4) = (1, 1)$. Setting $\lambda = \frac{1}{16}$, the regularized risk (1) can be written as (using $w$ instead of $\mathbf{w}$ as it is now a scalar):

$$\min_{w \in \mathbb{R}} J(w) = \frac{1}{32} w^2 + \frac{1}{2} \left[ 1 - \frac{w}{2} \right]_+ + \frac{1}{2} [1 - w]_+ . \qquad (10)$$

The minimizer of $J(w)$ is $w^* = 2$, which can be verified by the fact that 0 is in the subdifferential of $J$ at $w^* : 0 \in \partial J(2) = \left\{ \frac{2}{16} - \frac{1}{2}\frac{1}{2}\alpha : \alpha \in [0, 1] \right\}$. So $J(w^*) = \frac{1}{8}$. Choosing $w_0 = 0$, we have

**Theorem 2** $\lim_{k \to \infty} k \left( J(w_k) - J(w^*) \right) = \frac{1}{4}$, i.e. $J(w_k)$ *converges to* $J(w^*)$ *at* $1/k$ *rate.*

The proof relies on two lemmata. The first shows that the iterates generated by ls-bmrm on $J(w)$ satisfy the following recursive relations.

**Lemma 3** *For* $k \geq 1$, *the following recursive relations hold true*

$$w_{2k+1} = 2 + \frac{8\alpha_{2k-1,1}\left(w_{2k-1} - 4\alpha_{2k-1,1}\right)}{w_{2k-1}\left(w_{2k-1} + 4\alpha_{2k-1,1}\right)} > 2, \quad and \quad w_{2k} = 2 - \frac{8\alpha_{2k-1,1}}{w_{2k-1}} \in (1, 2). \quad (11)$$

$$\alpha_{2k+1,1} = \frac{w_{2k-1}^2 + 16\alpha_{2k-1,1}^2}{\left(w_{2k-1} + 4\alpha_{2k-1,1}\right)^2} \alpha_{2k-1,1}, \text{where } \alpha_{2k+1,1} \text{ is the first coordinate of } \boldsymbol{\alpha}_{2k+1}. \quad (12)$$

The proof is lengthy and is available at [7, Appendix B]. These recursive relations allow us to derive the convergence rate of $\alpha_{2k-1,1}$ and $w_k$ (see proof in [7, Appendix C]):

**Lemma 4** $\lim_{k\to\infty} k\alpha_{2k-1,1} = \frac{1}{4}$. *Combining with* (11), *we get* $\lim_{k\to\infty} k|2 - w_k| = 2$.

Now that $w_k$ approaches 2 at the rate of $O(1/k)$, it is finally straightforward to translate it into the rate at which $J(w_k)$ approaches $J(w^*)$. See the proof of Theorem 2 in [7, Appendix D].

## 4.2 Weak Lower Bounds for Solving Linear SVMs using qp-bmrm

Theorem 1 gives an upper bound on the convergence rate of qp-bmrm, assuming that $R_{\text{emp}}$ satisfies the assumption **A2**. In this section we further demonstrate that this $O(1/\epsilon)$ rate is also a WLB (hence tight) even when the $R_{\text{emp}}$ is specialized to SVM objectives satisfying **A2**.

Given $\epsilon > 0$, define $n = \lceil 1/\epsilon \rceil$ and construct a dataset $\{(\mathbf{x}_i, y_i)\}_{i=1}^n$ as $y_i = (-1)^i$ and $\mathbf{x}_i = (-1)^i (n\mathbf{e}_{i+1} + \sqrt{n}\mathbf{e}_1) \in \mathbb{R}^{n+1}$. Then the corresponding objective function (1) is

$$
J(\mathbf{w}) = \frac{\|\mathbf{w}\|^2}{2} + R_{\text{emp}}(\mathbf{w}), \text{ where } R_{\text{emp}}(\mathbf{w}) = \frac{1}{n}\sum_{i=1}^n [1 - y_i\langle\mathbf{w}, \mathbf{x}_i\rangle]_+ = \frac{1}{n}\sum_{i=1}^n [1 - \sqrt{n}w_1 - nw_{i+1}]_+.
$$
(13)

It is easy to see that the minimizer $\mathbf{w}^* = \frac{1}{2}(\frac{1}{\sqrt{n}}, \frac{1}{n}, \frac{1}{n}, \ldots, \frac{1}{n})^\top$ and $J(\mathbf{w}^*) = \frac{1}{4n}$. In fact, simply check that $y_i \langle \mathbf{w}^*, \mathbf{x}_i \rangle = 1$, so $\partial J(\mathbf{w}^*) = \left\{ \mathbf{w}^* - \left( \frac{1}{\sqrt{n}}\sum_{i=1}^n \alpha_i, \alpha_1, \ldots, \alpha_n \right)^\top : \alpha_i \in [0, 1] \right\}$, and setting all $\alpha_i = \frac{1}{2n}$ yields the subgradient $\mathbf{0}$. Our key result is the following theorem.

**Theorem 5** *Let* $\mathbf{w}_0 = (\frac{1}{\sqrt{n}}, 0, 0, \ldots)^\top$. *Suppose running* qp-bmrm *on the objective function* (13) *produces iterates* $\mathbf{w}_1, \ldots, \mathbf{w}_k, \ldots$. *Then it takes* qp-bmrm *at least* $\lfloor \frac{2}{3\epsilon} \rfloor$ *steps to find an $\epsilon$ accurate solution. Formally,*

$$
\min_{i\in[k]} J(\mathbf{w}_i) - J(\mathbf{w}^*) = \frac{1}{2k} + \frac{1}{4n} \text{ for all } k \in [n], \text{ hence } \min_{i\in[k]} J(\mathbf{w}_i) - J(\mathbf{w}^*) > \epsilon \text{ for all } k < \frac{2}{3\epsilon}.
$$

Indeed, after taking $n$ steps, $\mathbf{w}_n$ will cut a subgradient $\mathbf{a}_{n+1} = \mathbf{0}$ and $b_{n+1} = 0$, and then the minimizer of $J_{n+1}(\mathbf{w})$ gives exactly $\mathbf{w}^*$.

**Proof** Since $R_{\text{emp}}(\mathbf{w}_0) = 0$ and $\partial R_{\text{emp}}(\mathbf{w}_0) = \left\{ \frac{-1}{n}\sum_{i=1}^n \alpha_i y_i \mathbf{x}_i : \alpha_i \in [0, 1] \right\}$, we can choose

$$
\mathbf{a}_1 = -\frac{1}{n}y_1\mathbf{x}_1 = \left( -\frac{1}{\sqrt{n}}, -1, 0, \ldots \right)^\top, \quad b_1 = R_{\text{emp}}(\mathbf{w}_0) - \langle \mathbf{a}_1, \mathbf{w}_0 \rangle = 0 + \frac{1}{n} = \frac{1}{n}, \text{ and}
$$

$$
\mathbf{w}_1 = \underset{\mathbf{w}}{\text{argmin}} \left\{ \frac{1}{2}\|\mathbf{w}\|^2 - \frac{1}{\sqrt{n}}w_1 - w_2 + \frac{1}{n} \right\} = \left( \frac{1}{\sqrt{n}}, 1, 0, \ldots \right)^\top.
$$

In general, we claim that the $k$-th iterate $\mathbf{w}_k$ produced by qp-bmrm is given by

$$
\mathbf{w}_k = \left( \frac{1}{\sqrt{n}}, \overbrace{\frac{1}{k}, \ldots, \frac{1}{k}}^{k \text{ copies}}, 0, \ldots \right)^\top.
$$

We prove this claim by induction on $k$. Assume the claim holds true for steps $1, \ldots, k$, then it is easy to check that $R_{\text{emp}}(\mathbf{w}_k) = 0$ and $\partial R_{\text{emp}}(\mathbf{w}_k) = \left\{ \frac{-1}{n}\sum_{i=k+1}^n \alpha_i y_i \mathbf{x}_i : \alpha_i \in [0, 1] \right\}$. Thus we can again choose

$$
\mathbf{a}_{k+1} = -\frac{1}{n}y_{k+1}\mathbf{x}_{k+1}, \quad \text{and} \quad b_{k+1} = R_{\text{emp}}(\mathbf{w}_k) - \langle \mathbf{a}_{k+1}, \mathbf{w}_k \rangle = \frac{1}{n}, \text{ so}
$$

$$
\mathbf{w}_{k+1} = \underset{\mathbf{w}}{\text{argmin}} \left\{ \frac{1}{2}\|\mathbf{w}\|^2 + \max_{1\le i\le k+1} \{\langle \mathbf{a}_i, \mathbf{w}\rangle + b_i\} \right\} = \left( \frac{1}{\sqrt{n}}, \overbrace{\frac{1}{k+1}, \ldots, \frac{1}{k+1}}^{k+1 \text{ copies}}, 0, \ldots \right)^\top,
$$

which can be verified by checking that $\partial J_{k+1}(\mathbf{w}_{k+1}) = \left\{ \mathbf{w}_{k+1} + \sum_{i\in[k+1]} \alpha_i \mathbf{a}_i : \boldsymbol{\alpha} \in \Delta_{k+1} \right\} \ni \mathbf{0}$. All that remains is to observe that $J(\mathbf{w}_k) = \frac{1}{2k} + \frac{1}{2n}$ while $J(\mathbf{w}^*) = \frac{1}{4n}$ from which it follows that $J(\mathbf{w}_k) - J(\mathbf{w}^*) = \frac{1}{2k} + \frac{1}{4n}$ as claimed. ∎

As an aside, the subgradient of the $R_{\text{emp}}$ in (13) does have Euclidean norm $\sqrt{2n}$ at $\mathbf{w} = \mathbf{0}$. However, in the above run of qp-bmrm, $\partial R_{\text{emp}}(\mathbf{w}_0), \ldots, \partial R_{\text{emp}}(\mathbf{w}_n)$ always contains a subgradient with norm 1. So if we restrict the feasible region to $\{n^{-1/2}\} \times [0, \infty]^n$, then $J(\mathbf{w})$ does satisfy the assumption **A2** and the optimal solution does not change. This is essentially a local satisfaction of **A2**. In fact, having a bounded subgradient of $R_{\text{emp}}$ at all $\mathbf{w}_k$ is sufficient for qp-bmrm to converge at the rate in Theorem 1.

However when we assume **A1** which is more restrictive than **A2**, it remains an open question to determine whether the $O(1/\epsilon)$ rates are optimal for qp-bmrm on SVM objectives. Also left open is the SLB for qp-bmrm on SVMs.

### 4.3 Weak Lower Bounds for Optimizing $F_1$-score using qp-bmrm

$F_1$-score is defined by using the contingency table: $F_1(\bar{\mathbf{y}}, \mathbf{y}) := \frac{2a}{2a+b+c}$. Given $\epsilon > 0$, define $n = \lceil 1/\epsilon \rceil + 1$ and construct a dataset $\{(\mathbf{x}_i, y_i)\}_{i=1}^n$ as follows: $\mathbf{x}_i = -\frac{n}{2\sqrt{3}}\mathbf{e}_1 - \frac{n}{2}\mathbf{e}_{i+1} \in \mathbb{R}^{n+1}$ with $y_i = -1$ for all $i \in [n-1]$, and $\mathbf{x}_n = \frac{\sqrt{3}n}{2}\mathbf{e}_1 + \frac{n}{2}\mathbf{e}_{n+1} \in \mathbb{R}^{n+1}$ with $y_n = +1$. So there is only one positive training example. Then the corresponding objective function is

|  | $y=1$ | $y=-1$ |
|---|---|---|
| $\bar{y}=1$ | $a$ | $b$ |
| $\bar{y}=-1$ | $c$ | $d$ |

Contingency table.

$$J(\mathbf{w}) = \frac{1}{2}\|\mathbf{w}\|^2 + \max_{\bar{\mathbf{y}}}\left[1 - F_1(\mathbf{y}, \bar{\mathbf{y}}) + \frac{1}{n}\sum_{i=1}^n y_i \langle \mathbf{w}, \mathbf{x}_i \rangle (y_i\bar{y}_i - 1)\right]. \qquad (14)$$

**Theorem 6** *Let* $\mathbf{w}_0 = \frac{1}{\sqrt{3}}\mathbf{e}_1$. *Then* qp-bmrm *takes at least* $\lfloor \frac{1}{3\epsilon} \rfloor$ *steps to find an $\epsilon$ accurate solution.*

$$J(\mathbf{w}_k) - \min_{\mathbf{w}} J(\mathbf{w}) \geq \frac{1}{2}\left(\frac{1}{k} - \frac{1}{n-1}\right) \forall k \in [n-1], \text{ hence } \min_{i \in [k]} J(\mathbf{w}_i) - \min_{\mathbf{w}} J(\mathbf{w}) > \epsilon \, \forall k < \frac{1}{3\epsilon}.$$

**Proof** A rigorous proof can be found in [7, Appendix E], we provide a sketch here. The crux is to show

$$\mathbf{w}_k = \left(\frac{1}{\sqrt{3}}, \overbrace{\frac{1}{k}, \ldots, \frac{1}{k}}^{k \text{ copies}}, 0, \ldots\right)^\top \qquad \forall k \in [n-1]. \qquad (15)$$

We prove (15) by induction. Assume it holds for steps $1, \ldots, k$. Then at step $k+1$ we have

$$\frac{1}{n}y_i \langle \mathbf{w}_k, \mathbf{x}_i \rangle = \begin{cases} \frac{1}{6} + \frac{1}{2k} & \text{if } i \in [k] \\ \frac{1}{6} & \text{if } k+1 \leq i \leq n-1 \\ \frac{1}{2} & \text{if } i = n \end{cases}. \qquad (16)$$

For convenience, define the term in the $\max$ in (14) as

$$\Upsilon_k(\bar{\mathbf{y}}) := 1 - F_1(\mathbf{y}, \bar{\mathbf{y}}) + \frac{1}{n}\sum_{i=1}^n y_i \langle \mathbf{w}_k, \mathbf{x}_i \rangle (y_i\bar{y}_i - 1).$$

Then it is not hard to see that the following assignments of $\bar{\mathbf{y}}$ (among others) maximize $\Upsilon_k$: a) correct labeling, b) only misclassify the positive training example $\mathbf{x}_n$ (*i.e.*, $\bar{y}_n = -1$), c) only misclassify *one* negative training example in $\mathbf{x}_{k+1}, \ldots, \mathbf{x}_{n-1}$ into positive. And $\Upsilon_k$ equals 0 at all these assignments. For a proof, consider two cases. If $\bar{\mathbf{y}}$ misclassifies the positive training example, then $F_1(\mathbf{y}, \bar{\mathbf{y}}) = 0$ and by (16) we have

$$\Upsilon_k(\bar{\mathbf{y}}) = 1 - 0 + \frac{1}{n}\sum_{i=1}^{n-1} y_i \langle \mathbf{w}_k, \mathbf{x}_i \rangle (y_i\bar{y}_i - 1) + \frac{1}{2}(-1-1) = \frac{k+3}{6k}\sum_{i=1}^k (y_i\bar{y}_i - 1) + \frac{1}{6}\sum_{i=k+1}^{n-1}(y_i\bar{y}_i - 1) \leq 0.$$

Suppose $\bar{\mathbf{y}}$ correctly labels the positive example, but misclassifies $t_1$ examples in $\mathbf{x}_1, \ldots, \mathbf{x}_k$ and $t_2$ examples in $\mathbf{x}_{k+1}, \ldots, \mathbf{x}_{n-1}$ (into positive). Then $F_1(\mathbf{y}, \bar{\mathbf{y}}) = \frac{2}{2+t_1+t_2}$, and

$$\Upsilon_k(\bar{\mathbf{y}}) = 1 - \frac{2}{2+t_1+t_2} + \left(\frac{1}{6} + \frac{1}{2k}\right)\sum_{i=1}^k (y_i\bar{y}_i - 1) + \frac{1}{6}\sum_{i=k+1}^{n-1}(y_i\bar{y}_i - 1)$$

$$= \frac{t_1+t_2}{2+t_1+t_2} - \left(\frac{1}{3} + \frac{1}{k}\right)t_1 - \frac{1}{3}t_2 \leq \frac{t - t^2}{3(2+t)} \leq 0 \quad (t := t_1 + t_2).$$

So we can pick $\bar{\mathbf{y}}$ as $(\overbrace{-1,\ldots,-1}^{k\text{ copies}},+1,\overbrace{-1,\ldots,-1}^{n-k-1\text{ copies}},+1)^{\top}$ which only misclassifies $\mathbf{x}_{k+1}$, and get

$$\mathbf{a}_{k+1} = \frac{-2}{n}y_{k+1}\mathbf{x}_{k+1} = -\frac{1}{\sqrt{3}}\mathbf{e}_1 - \mathbf{e}_{k+2}, \quad b_{k+1} = R_{\mathrm{emp}}(\mathbf{w}_k) - \langle\mathbf{a}_{k+1},\mathbf{w}_k\rangle = 0 + \frac{1}{3} = \frac{1}{3},$$

$$\mathbf{w}_{k+1} = \operatorname*{argmin}_{\mathbf{w}} \overbrace{\frac{1}{2}\left\|\mathbf{w}\right\|^2 + \max_{i\in[k+1]}\left\{\langle\mathbf{a}_i,\mathbf{w}\rangle + b_i\right\}}^{:=J_{k+1}(\mathbf{w})} = \left(\frac{1}{\sqrt{3}},\overbrace{\frac{1}{k+1},\ldots,\frac{1}{k+1}}^{k+1\text{ copies}},0,\ldots\right)^{\top}.$$

which can be verified by $\partial J_{k+1}(\mathbf{w}_{k+1}) = \left\{\mathbf{w}_{k+1} + \sum_{i=1}^{k+1}\alpha_i\mathbf{a}_i : \boldsymbol{\alpha}\in\Delta_{k+1}\right\} \ni \mathbf{0}$ (just set all $\alpha_i = \frac{1}{k+1}$). So (15) holds for step $k+1$. End of induction.

All that remains is to observe that $J(\mathbf{w}_k) = \frac{1}{2}(\frac{1}{3}+\frac{1}{k})$ while $\min_{\mathbf{w}} J(\mathbf{w}) \leq J(\mathbf{w}_{n-1}) = \frac{1}{2}(\frac{1}{3}+\frac{1}{n-1})$ from which it follows that $J(\mathbf{w}_k) - \min_{\mathbf{w}} J(\mathbf{w}) \geq \frac{1}{2}(\frac{1}{k}-\frac{1}{n-1})$ as claimed in Theorem 6. ∎

# 5   An $O(nd/\sqrt{\epsilon})$ Algorithm for Training Binary Linear SVMs

The lower bounds we proved above show that CPM such as BMRM require $\Omega(1/\epsilon)$ iterations to converge. We now show that this is an inherent limitation of CPM and not an artifact of the problem. To demonstrate this, we will show that one can devise an algorithm for problems (1) and (2) which will converge in $O(1/\sqrt{\epsilon})$ iterations. The key difficulty stems from the non-smoothness of the objective function, which renders second and higher order algorithms such as L-BFGS inapplicable. However, thanks to [7, Theorem 7 in Appendix A], the Fenchel dual of (1) is a convex smooth function with a Lipschitz continuous gradient, which are easy to optimize.

To formalize the idea of using the Fenchel dual, we can abstract from the objectives (1) and (2) a *composite* form of objective functions used in machine learning with linear models:

$$\min_{\mathbf{w}\in Q_1} J(\mathbf{w}) = f(\mathbf{w}) + g^{\star}(A\mathbf{w}), \quad \text{where } Q_1 \text{ is a closed convex set.} \tag{17}$$

Here, $f(\mathbf{w})$ is a strongly convex function corresponding to the regularizer, $A\mathbf{w}$ stands for the output of a linear model, and $g^{\star}$ encodes the empirical risk measuring the discrepancy between the correct labels and the output of the linear model. Let the domain of $g$ be $Q_2$. It is well known that [*e.g.* 8, Theorem 3.3.5] under some mild constraint qualifications, the adjoint form of $J(\mathbf{w})$:

$$D(\boldsymbol{\alpha}) = -g(\boldsymbol{\alpha}) - f^{\star}(-A^{\top}\boldsymbol{\alpha}), \quad \boldsymbol{\alpha}\in Q_2 \tag{18}$$

satisfies $J(\mathbf{w}) \geq D(\boldsymbol{\alpha})$ and $\inf_{\mathbf{w}\in Q_1} J(\mathbf{w}) = \sup_{\boldsymbol{\alpha}\in Q_2} D(\boldsymbol{\alpha})$.

**Example 1: binary SVMs with bias.**   Let $A := -YX^{\top}$ where $Y := \operatorname{diag}(y_1,\ldots,y_n)$ and $X := (\mathbf{x}_1,\ldots,\mathbf{x}_n)$, $f(\mathbf{w}) = \frac{\lambda}{2}\left\|\mathbf{w}\right\|^2$, $g^{\star}(\mathbf{u}) = \min_{b\in\mathbb{R}}\frac{1}{n}\sum_{i=1}^{n}\left[1 + u_i - y_i b\right]_{+}$ which corresponds to $g(\boldsymbol{\alpha}) = -\sum_i \alpha_i$. Then the adjoint form turns out to be the well known SVM dual objective function:

$$D(\boldsymbol{\alpha}) = \sum_i \alpha_i - \frac{1}{2\lambda}\boldsymbol{\alpha}^{\top}YX^{\top}XY\boldsymbol{\alpha}, \quad \boldsymbol{\alpha}\in Q_2 = \left\{\boldsymbol{\alpha}\in[0,n^{-1}]^n : \sum_i y_i\alpha_i = 0\right\}. \tag{19}$$

**Example 2: multivariate scores.**   Denote $A$ as a $2^n$-by-$d$ matrix where the $\bar{\mathbf{y}}$-th row is $\sum_{i=1}^{n}\mathbf{x}_i^{\top}(\bar{y}_i - y_i)$ for each $\bar{\mathbf{y}}\in\{-1,+1\}^n$, $f(\mathbf{w}) = \frac{\lambda}{2}\left\|\mathbf{w}\right\|^2$, $g^{\star}(\mathbf{u}) = \max_{\bar{\mathbf{y}}}\left[\Delta(\mathbf{y},\bar{\mathbf{y}}) + \frac{1}{n}u_{\bar{\mathbf{y}}}\right]$ which corresponds to $g(\boldsymbol{\alpha}) = -n\sum_{\bar{\mathbf{y}}}\Delta(\mathbf{y},\bar{\mathbf{y}})\alpha_{\bar{\mathbf{y}}}$, we recover the primal objective (2) for multivariate performance measure. Its adjoint form is

$$D(\boldsymbol{\alpha}) = -\frac{1}{2\lambda}\boldsymbol{\alpha}^{\top}AA^{\top}\boldsymbol{\alpha} + n\sum_{\bar{\mathbf{y}}}\Delta(\mathbf{y},\bar{\mathbf{y}})\alpha_{\bar{\mathbf{y}}}, \ \ \boldsymbol{\alpha}\in Q_2 = \left\{\boldsymbol{\alpha}\in[0,n^{-1}]^{2^n} : \sum_{\bar{\mathbf{y}}}\alpha_{\bar{\mathbf{y}}} = \frac{1}{n}\right\}. \tag{20}$$

In a series of papers [6, 9, 10], Nesterov developed *optimal* gradient based methods for minimizing the composite objectives with primal (17) and adjoint (18). A sequence of $\mathbf{w}_k$ and $\boldsymbol{\alpha}_k$ is produced such that under assumption **A1** the duality gap $J(\mathbf{w}_k) - D(\boldsymbol{\alpha}_k)$ is reduced to less than $\epsilon$ after at most $k = O(1/\sqrt{\epsilon})$ steps. We refer the readers to [9, 11] for details.

## 5.1 Efficient Projections in Training SV Models with Optimal Gradient Methods

However, applying Nesterov's algorithm is challenging, because it requires an *efficient* subroutine for computing projections onto the set of constraints $Q_2$. This projection can be either an Euclidean projection or a Bregman projection.

**Example 1: binary SVMs with bias.** In this case we need to compute the Euclidean projection to $Q_2$ defined by (19), which entails solving a Quadratic Programming problem with a diagonal Hessian, many box constraints, and a single equality constraint. We present an $O(n)$ algorithm for this task in [11, Section 5.5.1]. Plugging this into the algorithm described in [9] and noting that all intermediate steps of the algorithm can be computed in $O(nd)$ time directly yield a $O(nd/\sqrt{\epsilon})$ algorithm. More detailed description of the algorithm is available in [11].

**Example 2: multivariate scores.** Since the dimension of $Q_2$ in (20) is exponentially large in $n$, Euclidean projection is intractable and we resort to Bregman projection. Given a differentiable convex function $F$ on $Q_2$, a point $\boldsymbol{\alpha}$, and a direction $\mathbf{g}$, we can define the Bregman projection as:
$$V(\boldsymbol{\alpha}, \mathbf{g}) := \underset{\bar{\boldsymbol{\alpha}} \in Q_2}{\operatorname{argmin}} F(\bar{\boldsymbol{\alpha}}) - \langle \nabla F(\boldsymbol{\alpha}) - \mathbf{g}, \bar{\boldsymbol{\alpha}} \rangle.$$
Scaling up $\boldsymbol{\alpha}$ by a factor of $n$, we can choose $F(\boldsymbol{\alpha})$ as the negative entropy $F(\boldsymbol{\alpha}) = -\sum_i \alpha_i \log \alpha_i$. Then the application of the algorithm in [9] will endow a distribution over all possible labelings:
$$p(\bar{\mathbf{y}}; \mathbf{w}) \propto \exp\left( c\Delta(\bar{\mathbf{y}}, \mathbf{y}) + \sum_i a_i \langle \mathbf{x}_i, \mathbf{w} \rangle \bar{y}_i \right), \quad \text{where } c \text{ and } a_i \text{ are constant scalars.} \quad (21)$$
The solver will request the expectation $\mathbb{E}_{\bar{\mathbf{y}}}\left[\sum_i a_i \mathbf{x}_i \bar{y}_i\right]$ which in turn requires that marginal distribution of $p(\bar{y}_i)$. This is not as straightforward as in graphical models because $\Delta(\bar{\mathbf{y}}, \mathbf{y})$ may not decompose. Fortunately, for multivariate scores defined by contingency tables, it is possible to compute the marginals in $O(n^2)$ time by using dynamic programming, and this cost is similar to the algorithm proposed by [3]. The detail of the dynamic programming is given in [11, Section 5.4].

## 6 Outlook and Conclusion

CPM are widely employed in machine learning especially in the context of structured prediction [12]. While upper bounds on their rates of convergence were known, lower bounds were not studied before. In this paper we set out to fill this gap by exhibiting counter examples in binary classification on which CPM require $\Omega(1/\epsilon)$ iterations. Our examples are substantially different from the one in [13] which requires an increasing number of classes. The $\Omega(1/\epsilon)$ lower bound is a fundamental limitation of these algorithms and not an artifact of the problem. We show this by devising an $O(1/\sqrt{\epsilon})$ algorithm borrowing techniques from [9]. However, this algorithm assumes that the dataset is contained in a ball of bounded radius (assumption **A1** Section 1). Devising a $O(1/\sqrt{\epsilon})$ algorithm under the less restrictive assumption **A2** remains an open problem.

It is important to note that the linear time algorithm in [11, Section 5.5.1] is the key to obtaining a $O(nd/\sqrt{\epsilon})$ computational complexity for binary SVMs with bias mentioned in Section 5.1. However, this method has been rediscovered independently by many authors (including us), with the earliest known reference to the best of our knowledge being [14] in 1990. Some recent work in optimization [15] has focused on improving the practical performance, while in machine learning [16] gave an expected linear time algorithm via randomized median finding.

Choosing an optimizer for a given machine learning task is a trade-off between a number of potentially conflicting requirements. CPM are one popular choice but there are others. If one is interested in classification accuracy alone, without requiring deterministic guarantees, then online to batch conversion techniques combined with stochastic subgradient descent are a good choice [17]. While the dependence on $\epsilon$ is still $\Omega(1/\epsilon)$ or worse [18], one gets bounds independent of $n$. However, as we pointed out earlier, these algorithms are applicable only when the empirical risk decomposes over the examples.

On the other hand, one can employ coordinate descent in the dual as is done in the Sequential Minimal Optimization (SMO) algorithm of [19]. However, as [20] show, if the kernel matrix obtained by stacking $\mathbf{x}_i$ into a matrix $X$ and $X^\top X$ is not strictly positive definite, then SMO requires $O(n/\epsilon)$ iterations with each iteration costing $O(nd)$ effort. However, when the kernel matrix is strictly positive definite, then one can obtain an $O(n^2 \log(1/\epsilon))$ bound on the number of iterations, which has better dependence on $\epsilon$, but is prohibitively expensive for large $n$. Even better dependence on $\epsilon$ can be achieved by using interior point methods [21] which require only $O(\log(\log(1/\epsilon))$ iterations, but the time complexity per iteration is $O(\min\{n^2 d, d^2 n\})$.

## Footnotes

[1]In this paper we use the term cutting plane methods to denote specialized solvers employed in machine learning. While clearly related, they must not be confused with cutting plane methods used in optimization.

[2]Because of the specialized nature of these solvers, lower bounds for *general* convex optimizers such as those studied by Nesterov [4] and Nemirovski and Yudin [5] do not apply.

[3] The initial point also matters, as in the best case we can just start from the optimal solution. Thus the quantity of interest is actually $T(\epsilon; f, A) := \max_{\mathbf{w}_0} \min\{k : f(\mathbf{w}_k) - \min_{\mathbf{w}} f(\mathbf{w}) \le \epsilon$, starting point being $\mathbf{w}_0\}$. However, without loss of generality we assume some pre-specified way of initialization.

# References

[1] T. Joachims. Training linear SVMs in linear time. In *Proc. ACM Conf. Knowledge Discovery and Data Mining (KDD)*, pages 217–226, 2006.

[2] C. H. Teo, S. V. N. Vishwanthan, A. J. Smola, and Q. V. Le. Bundle methods for regularized risk minimization. *J. Mach. Learn. Res.*, 11:311–365, January 2010.

[3] T. Joachims. A support vector method for multivariate performance measures. In *Proc. Intl. Conf. Machine Learning*, pages 377–384, 2005.

[4] Y. Nesterov. *Introductory Lectures On Convex Optimization: A Basic Course*. Springer, 2003.

[5] A. Nemirovski and D. Yudin. *Problem Complexity and Method Efficiency in Optimization*. John Wiley and Sons, 1983.

[6] Y. Nesterov. A method for unconstrained convex minimization problem with the rate of convergence $O(1/k^2)$. *Soviet Math. Docl.*, 269:543–547, 1983.

[7] Xinhua Zhang, Ankan Saha, and S.V.N. Vishwanathan. Lower bounds on rate of convergence of cutting plane methods (long version). Technical report, 2010. http://www.stat.purdue.edu/~vishy/papers/ZhaSahVis10_long.pdf.

[8] J. M. Borwein and A. S. Lewis. *Convex Analysis and Nonlinear Optimization: Theory and Examples*. CMS books in Mathematics. Canadian Mathematical Society, 2000.

[9] Y. Nesterov. Excessive gap technique in nonsmooth convex minimization. *SIAM Journal on Optimization*, 16(1):235–249, 2005. ISSN 1052-6234.

[10] Y. Nesterov. Gradient methods for minimizing composite objective function. Technical Report 76, CORE Discussion Paper, UCL, 2007.

[11] Xinhua Zhang, Ankan Saha, and S.V.N. Vishwanathan. Regularized risk minimization by Nesterov's accelerated gradient methods: Algorithmic extensions and empirical studies. Technical report arXiv:1011.0472, 2010. http://arxiv.org/abs/1011.0472.

[12] I. Tsochantaridis, T. Joachims, T. Hofmann, and Y. Altun. Large margin methods for structured and interdependent output variables. *J. Mach. Learn. Res.*, 6:1453–1484, 2005.

[13] T. Joachims, T. Finley, and C.N.J̃. Yu. Cutting-plane training of structural SVMs. *Machine Learning Journal*, 77(1):27–59, 2009.

[14] P. M. Pardalos and N. Kovoor. An algorithm for singly constrained class of quadratic programs subject to upper and lower bounds. *Mathematical Programming*, 46:321–328, 1990.

[15] Y.-H. Dai and R. Fletcher. New algorithms for singly linearly constrained quadratic programs subject to lower and upper bounds. *Mathematical Programming: Series A and B archive*, 106 (3):403–421, 2006.

[16] J. Duchi, S. Shalev-Shwartz, Y. Singer, and T. Chandra. Efficient projections onto the $\ell_1$-ball for learning in high dimensions. In *Proc. Intl. Conf. Machine Learning*, pages 272–279, 2008.

[17] S. Shalev-Shwartz, Y. Singer, and N. Srebro. Pegasos: Primal estimated sub-gradient solver for SVM. In *Proc. Intl. Conf. Machine Learning*, pages 807–814, 2007.

[18] A. Agarwal, P. L. Bartlett, P. Ravikumar, and M. Wainwright. Information-theoretic lower bounds on the oracle complexity of convex optimization. In *Neural Information Processing Systems*, pages 1–9, 2009.

[19] J. C. Platt. Sequential minimal optimization: A fast algorithm for training support vector machines. Technical Report MSR-TR-98-14, Microsoft Research, 1998.

[20] N. List and H. U. Simon. SVM-optimization and steepest-descent line search. In S. Dasgupta and A. Klivans, editors, *Proc. Annual Conf. Computational Learning Theory*, 2009.

[21] M. C. Ferris and T. S. Munson. Interior-point methods for massive support vector machines. *SIAM Journal on Optimization*, 13(3):783–804, 2002.

